# Why did TD-Gammon Work?

**Jordan B. Pollack & Alan D. Blair**
Computer Science Department
Brandeis University
Waltham, MA 02254
{pollack,blair}@cs.brandeis.edu

## Abstract

Although TD-Gammon is one of the major successes in machine learning, it has not led to similar impressive breakthroughs in temporal difference learning for other applications or even other games. We were able to replicate some of the success of TD-Gammon, developing a competitive evaluation function on a 4000 parameter feed-forward neural network, without using back-propagation, reinforcement or temporal difference learning methods. Instead we apply simple hill-climbing in a relative fitness environment. These results and further analysis suggest that the surprising success of Tesauro's program had more to do with the co-evolutionary structure of the learning task and the dynamics of the backgammon game itself.

## 1 INTRODUCTION

It took great *chutzpah* for Gerald Tesauro to start wasting computer cycles on temporal difference learning in the game of Backgammon (Tesauro, 1992). Letting a machine learning program play itself in the hopes of becoming an expert, indeed! After all, the dream of computers mastering a domain by self-play or "introspection" had been around since the early days of AI, forming part of Samuel's checker player (Samuel, 1959) and used in Donald Michie's MENACE tic-tac-toe learner (Michie, 1961). However such self-conditioning systems, with weak or non-existent internal representations, had generally been fraught with problems of scale and abandoned by the field of AI. Moreover, self-playing learners usually develop eccentric and brittle strategies which allow them to draw each other, yet play poorly against humans and other programs.

Yet Tesauro's 1992 result showed that this self-play approach could be powerful, and after some refinement and millions of iterations of self-play, his TD-Gammon program has become one of the best backgammon players in the world (Tesauro, 1995). His derived weights are viewed by his corporation as significant enough intellectual property to keep as a trade secret, except to leverage sales of their minority operating system (International Business Machines, 1995). Others have replicated this TD result both for research purposes (Boyan, 1992) and commercial purposes.

With respect to the goal of a self-organizing learning machine which starts from a minimal specification and rises to great sophistication, TD-Gammon stands alone. How is its success to be understood, explained, and replicated in other domains? Is TD-Gammon unbridled good news about the reinforcement learning method?

Our hypothesis is that the success of TD-gammon is not due to the back-propagation, reinforcement, or temporal-difference technologies, but to an inherent bias from the dynamics of the game of backgammon, and the co-evolutionary setup of the training, by which the task dynamically changes as the learning progresses. We test this hypothesis by using a much simpler co-evolutionary learning method for backgammon - namely hill-climbing.

## 2 SETUP

We use a standard feedforward neural network with two layers and the sigmoid function, set up in the same fashion as Tesauro with 4 units to represent the number of each player's pieces on each of the 24 points, plus 2 units each to indicate how many are on the bar and off the board. In addition, we added one more unit which reports whether or not the game has reached the endgame or "race" situation, making a total of 197 input units. These are fully connected to 20 hidden units, which are then connected to one output unit that judges the position. Including bias on the hidden units, this makes a total of 3980 weights. The game is played by generating all legal moves, converting them into the proper network input, and picking the position judged as best by the network. We started with all weights set to zero.

Our initial algorithm was hillclimbing:

1. add gaussian noise to the weights
2. play the network against the mutant for a number of games
3. if the mutant wins more than half the games, select it for the next generation.

The noise was set so each step would have a 0.05 RMS distance (which is the euclidean distance divided by $\sqrt{3980}$).

Surprisingly, this worked reasonably well! The networks so evolved improved rapidly at first, but then sank into mediocrity. The problem we perceived is that comparing two close backgammon players is like tossing a biased coin repeatedly: it may take dozens or even hundreds of games to find out for sure which of them is better. Replacing a well-tested champion is dangerous without enough information to prove the challenger is really a better player and not just a lucky novice. Rather than burden the system with so much computation, we instead introduced the following modifications to the algorithm to avoid this "Buster Douglas Effect":

Firstly, the games are played in pairs, with the order of play reversed and the same random seed used to generate the dice rolls for both games. This washes out some of the unfairness due to the dice rolls when the two networks are very close - in particular, if they were identical, the result would always be one win each. Secondly, when the challenger wins the contest, rather than just replacing the champion by the challenger, we instead make only a small adjustment in that direction:

$$\text{champion} := 0.95 * \text{champion} + 0.05 * \text{challenger}$$

This idea, similar to the "inertia" term in back-propagation, was introduced on the assumption that small changes in weights would lead to small changes in decision-making by the evaluation function. So, by preserving most of the current champion's decisions, we would be less likely to have a catastrophic replacement of the champion by a lucky novice challenger.

In the initial stages of evolution, two pairs of parallel games were played and the challenger was required to win 3 out of 4 of these games.

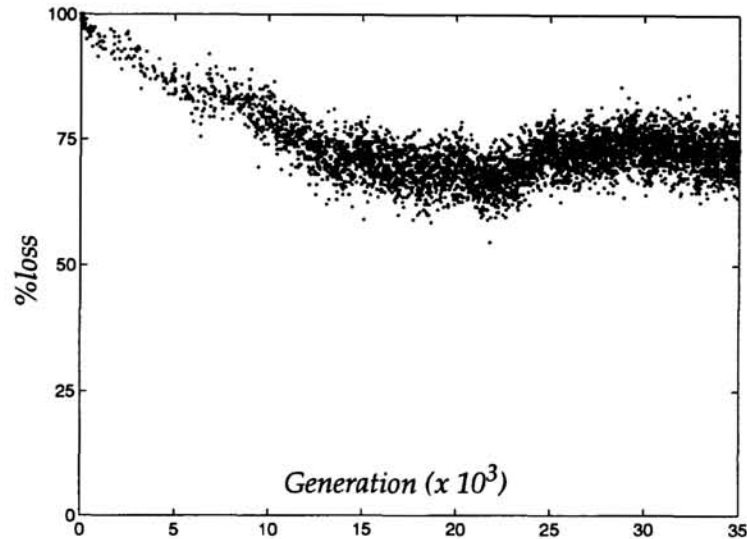

Figure 1. Percentage of losses of our first 35,000 generation players against PUBEVAL. Each match consisted of 200 games.

Figure 1 shows the first 35,000 players rated against PUBEVAL, a strong public-domain player trained by Tesauro using human expert preferences. There are three things to note: (1) the percentage of losses against PUBEVAL falls from 100% to about 67% by 20,000 generations, (2) the frequency of successful challengers increases over time as the player improves, and (3) there are epochs (e.g. starting at 20,000) where the performance against PUBEVAL begins to falter. The first fact shows that our simple self-playing hill-climber is capable of learning. The second fact is quite counter-intuitive - we expected that as the player improved, it would be harder to challenge it! This is true with respect to a uniform sampling of the 4000 dimensional weight space, but not true for a sampling in the neighborhood of a given player: once the player is in a good part of weight space, small changes in weights can lead to mostly similar strategies, ones which make mostly the same moves in the same situations. However, because of the few games we were using to determine relative fitness, this increased frequency of change allows the system to drift, which may account for the subsequent degrading of performance.

To counteract the drift, we decided to change the rules of engagement as the evolution proceeds according to the following "annealing schedule": after 10,000 generations, the number of games that the challenger is required to win was increased from 3 out of 4 to 5 out of 6; after 70,000 generations, it was further increased to 7 out of 8. The numbers 10,000 and 70,000 were chosen on an ad hoc basis from observing the frequency of successful challenges.

After 100,000 games, we have developed a surprisingly strong player, capable of winning 40% of the games against PUBEVAL. The networks were sampled every 100 generations in order to test their performance. Networks at generation 1,000, 10,000 and 100,000 were extracted and used as benchmarks. Figure 2 shows the percentage of losses of the sampled players against the three benchmark networks. Note that the three curves cross the 50% line at 1, 10, and 100, respectively and show a general improvement over time.

The end-game of backgammon, called the "bear-off," can be used as another yardstick of the progress of learning. The bear-off occurs when all of a player's pieces are in the player's home, or first 6 points, and then the dice rolls can be used to remove pieces.We set up a racing board with two pieces on each player's 1 through 7 point and one piece on the 8 point, and played a player against itself 200 games, averaging the number of rolls. We found a monotonic improvement, from 22 to less then 19 rolls, over the 100k generations. PUBEVAL scored 16.6 on this task.

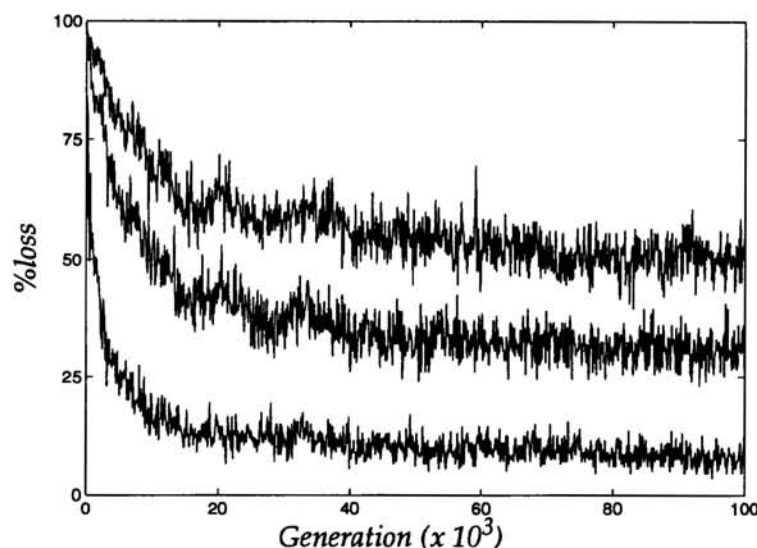

Figure 2. Percentage of losses against benchmark networks at generation 1,000 [lower], 10,000 [middle] and 100,000 [upper].

## 3 DISCUSSION

### 3.1 Machine Learning and Evolution

We believe that our evidence of success in learning backgammon using simple hillclimbing indicates that the reinforcement and temporal difference methodology used by Tesauro in TD-gammon was non-essential for its success. Rather, the success came from the setup of co-evolutionary self-play biased by the dynamics of backgammon. Our result is thus similar to the bias found by Mitchell, Crutchfield & Graber in Packard's evolution of cellular automata to the "edge of chaos"(Packard, 1988, Mitchell et al., 1993).

TD-Gammon is a major milestone for a kind of evolutionary machine learning in which the initial specification of model is far simpler than expected because the learning environment is specified implicitly, and emerges as a result of the *co-evolution* between a learning system and its training environment: The learner is embedded in an environment which responds to its own improvements in a never-ending spiral. While this effect has been seen in population models, it is completely unexpected for a "1+1" hillclimbing evolution.

Co-evolution was explored by Hillis (Hillis, 1992) on the sorting problem, by Angeline & Pollack (Angeline and Pollack, 1994) on genetically programmed tic-tac-toe players, on predator/prey games, e.g. (Cliff and Miller, 1995, Reynolds, 1994), and by Juille & Pollack on the intertwined spirals problem (Juille and Pollack, 1995). Rosin & Belew applied competitive fitness to several games (Rosin and Belew, 1995). However, besides Tesauro's TD-Gammon, which has not to date been viewed as an instance of co-evolutionary learning, Sims' artificial robot game (Sims, 1994) is the only other domain as complex as Backgammon to have had substantial success.

### 3.2 Learnability and Unlearnability

Learnability can be formally defined as a time constraint over a search space. How hard is it to randomly pick 4000 floating-point weights to make a good backgammon evaluator? It is simply impossible. How hard is it to find weights better than the current set? Initially, when all weights are random, it is quite easy. As the playing improves, we would expect it to get harder and harder, perhaps similar to the probability of a tornado constructing a 747 out of a junkyard. However, if we search in the neighborhood of the current weights, we will find many players which make mostly the same moves but which can capitalize on each other's slightly different choices and exposed weaknesses in a tournament.

Although the setting of parameters in our initial runs involved some guesswork, now that we have a large set of "players" to examine, we can try to understand the phenomenon. Taking the 1000th, 10,000th, and 100,000th champions from our run, we sampled random players in their neighborhoods at different RMS distances to find out how likely is it to find a winning challenger. We took 1000 random neighbors at each of 11 different RMS distances, and played them 8 games against the corresponding champion. Figure 3 plots

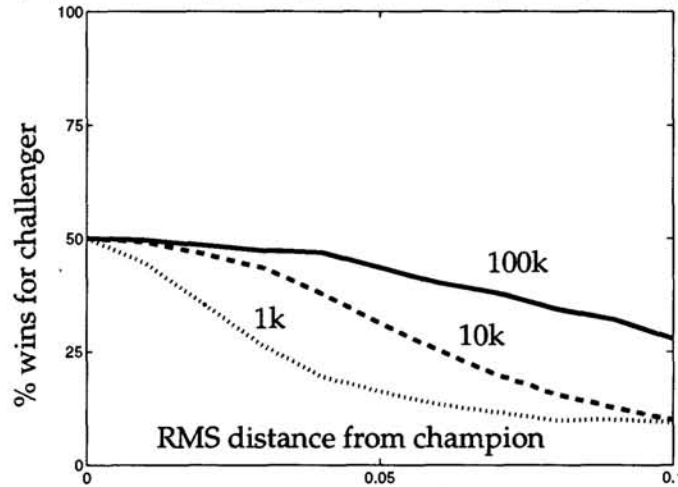

Figure 3. Distance versus probability of random challenger winning against champions at generation 1,000, 10,000 and 100,000.

the average number of games won against the three champions in the range of neighborhoods. This graph demonstrates that as the players improve over time, the probability of finding good challengers in their neighborhood **increases**. This accounts for why the frequency of successful challenges goes up. Each successive challenger is only required to take the small step of changing a few moves of the champion in order to beat it. Therefore, under co-evolution what was apparently unlearnable becomes learnable as we convert from a single question to a continuous stream of questions, each one dependent on the previous answer.

### 3.3 Avoiding Mediocre Stable States

In general, the problem with learning through self-play is that the player could keep playing the same kinds of games over and over, only exploring some narrow region of the strategy space, missing out on critical areas of the game where it could then be vulnerable to other programs or human experts. Such a learning system might declare success when in reality it has simply converged to a "mediocre stable state" of continual draws or a long term cooperation which merely mimics competition. Such a state can arise in human education systems, where the student gets all the answers right and rewards the teacher with positive feedback for not asking harder questions.

The problem is particularly prevalent in self-play for deterministic games such as chess or tic-tac-toe. We have worked on using a population to get around it (Angeline and Pollack, 1994). Schraudolph et al., 1994 added non-determinism to the game of Go by choosing moves according to the Boltzmann distribution of statistical mechanics. Others, such as Fogel, 1993, expanded exploration by forcing initial moves. Epstein, 1994, has studied a mix of training using self-play, random testing, and playing against an expert in order to better understand this phenomenon.

We are not suggesting that 1+1 hillclimbing is an advanced machine learning technique which others should bring to many tasks. Without internal cognition about an opponent's behavior, co-evolution usually requires a population. Therefore, there must be something about the dynamics of backgammon itself which is helpful because it permitted both TD

learning and hill-climbing to succeed where they would clearly fail on other tasks and in other games of this scale. If we can understand why the backgammon domain led to successful acquisition of expert strategies from random initial conditions, we might be able to re-cast other domains in its image.

Tesauro, 1992 pointed out some of the features of Backgammon that make it suitable for approaches involving self-play and random initial conditions. Unlike chess, a draw is impossible and a game played by an untrained network making random moves will eventually terminate (though it may take much longer than a game between competent players). Moreover the randomness of the dice rolls leads self-play into a much larger part of the search space than it would be likely to explore in a deterministic game.

We believe it is not simply the dice rolls which overcome the problems of self-learning. Others have tried to add randomness to deterministic games and have not generally met with success. There is something critical about the dynamics of backgammon which sets its apart from other games with random elements like Monopoly. Namely, that the outcome of the game continues to be uncertain until all contact is broken and one side has a clear advantage. What many observers find exciting about backgammon, and what helps a novice sometimes overcome an expert, is the number of situations where one dice roll, or an improbable sequence, can dramatically reverse which player is expected to win.

A learning system can be viewed as a meta-game between teacher and student, which are identical in a self-play situation. The teacher's goal is to expose the student's mistakes, while the student's goal is to placate the teacher and avoid such exposure. A mediocre stable state for a self-learning system can be seen as an equilibrium situation in this meta-game. A player which learns to repeatedly draw itself will have found a meta-game equilibrium and stop learning. If draws are not allowed, it may still be possible for a self-playing learner to collude with itself - to simulate competition while actually cooperating (Angeline, 1994). For example, if slightly suboptimal moves would allow a player to "throw" a game, a player under self-play could find a meta-game equilibrium by alternately throwing games to itself! Our hypothesis is that the dynamics of backgammon discussed above actively prevent this sort of collusion from forming in the meta-game of self-learning.

## 4 CONCLUSIONS

Tesauro's 1992 result beat Sun's Gammontool and achieved parity against his own Neurogammon 1.0, trained on expert knowledge. Neither of these is available. Following the 1992 paper on TD-learning, he incorporated a number of hand-crafted expert-knowledge features, eventually producing a network which achieved world master level play (Tesauro, 1995). These features included concepts like existence of a prime, probability of blots being hit, and probability of escape from behind the opponent's barrier. Our best players win about 45% against PUBEVAL which was trained using "comparison training"(Tesauro, 1989). Therefore we believe our players achieve approximately the same power as Tesauro's 1992 results, without any advanced learning algorithms. We do not claim that our 100,000 generation player is as good as TD-Gammon, ready to challenge the best humans, just that it is surprisingly good considering its humble origins from hill-climbing with a relative fitness measure. Tuning our parameters or adding more input features would make more powerful players, but that is not the point of this study.

TD-Gammon remains a tremendous success in Machine Learning, but the causes for its success have not been well understood. Replicating some of TD-Gammon's success under a much simpler learning paradigm, we find that the primary cause for success must be the dynamics of backgammon combined with the power of co-evolutionary learning. If we can isolate the features of the backgammon domain which enable evolutionary learning to work so well, it may lead to a better understanding of the conditions necessary, in general, for complex self-organization.

## Acknowledgments

This work is supported by ONR grant N00014-96-1-0418 and a Krasnow Foundation Postdoctoral fellowship. Thanks to Gerry Tesauro for providing PUBEVAL and subsequent means to calibrate it, Jack Laurence and Pablo Funes for development of the WWW front end to our evolved player. Interested players can challenge our evolved network using a web browser through our home page at: **http://www.demo.cs.brandeis.edu**

## References

Angeline, P. J. (1994). An alternate interpretation of the iterated prisoner's dilemma and the evolution of non-mutual cooperation. In Brooks, R. and Maes, P., editors, *Proceedings 4th Artificial Life Conference*, pages 353–358. MIT Press.

Angeline, P. J. and Pollack, J. B. (1994). Competitive environments evolve better solutions for complex tasks. In Forrest, S., editor, *Genetic Algorithms: Proceedings of the Fifth Inter national Conference*.

Boyan, J. A. (1992). Modular neural networks for learning context-dependent game strategies. Master's thesis, Computer Speech and Language Processing, Cambridge University.

Cliff, D. and Miller, G. (1995). Tracking the red queen: Measurements of adaptive progress in co-evolutionary simulations. In *Third European Conference on Artificial Life*, pages 200–218.

Hillis, D. (1992). Co-evolving parasites improves simulated evolution as an optimization procedure. In C. Langton, C. Taylor, J. F. and Rasmussen, S., editors, *Artificial Life II*. Addison-Wesley, Reading, MA.

International Business Machines (Sept. 12, 1995). IBM's family funpak for os/2 warp hits retail shelves.

Juille, H. and Pollack, J. (1995). Massively parallel genetic programming. In Angeline, P. and Kinnear, K., editors, *Advances in Genetic Programming II*. MIT Press, Cambridge.

Michie, D. (1961). Trial and error. In *Science Survey, part 2*, pages 129–145. Penguin.

Mitchell, M., Hraber, P. T., and Crutchfield, J. P. (1993). Revisiting the edge of chaos: Evolving cellular automata to perform computations. *Complex Systems*, 7.

Packard, N. (1988). Adaptation towards the edge of chaos. In Kelso, J. A. S., Mandell, A. J., and Shlesinger, M. F., editors, *Dynamic patterns in complex systems*, pages 293–301. World Scientific.

Reynolds, C. (1994). Competition, coevolution, and the game of tag. In *Proceedings 4th Artificial Life Conference*. MIT Press.

Rosin, C. D. and Belew, R. K. (1995). Methods for competitive co-evolution: finding opponents worth beating. In *Proceedings of the 6th international conference on Genetic Algorithms*, pages 373–380. Morgan Kaufman.

Samuel, A. L. (1959). some studies of machine learning using the game of checkers. *IBM Joural of Research and Development*.

Sims, K. (1994). Evolving 3d morphology and behavior by competition. In Brooks, R. and Maes, P., editors, *Proceedings 4th Artificial Life Conference*. MIT Press.

Tesauro, G. (1989). Connectionist learning of expert preferences by comparison training. In Touretzky, D., editor, *Advances in Neural Information Processing Systems*, volume 1, pages 99–106, Denver 1988. Morgan Kaufmann, San Mateo.

Tesauro, G. (1992). Practical issues in temporal difference learning. *Machine Learning*, 8:257–277.

Tesauro, G. (1995). Temporal difference learning and td-gammon. *Communications of the ACM*, 38(3):58–68.